# An aVLSI cricket ear model

**André van Schaik**[*]
The University of Sydney
NSW 2006, AUSTRALIA
*andre@ee.usyd.edu.au*

**Richard Reeve**[+]
University of Edinburgh
Edinburgh, UK
*richardr@inf.ed.ac.uk*

**Craig Jin**[*]
*craig@ee.usyd.edu.au*

**Tara Hamilton**[*]
*tara@ee.usyd.edu.au*

## Abstract

Female crickets can locate males by phonotaxis to the mating song they produce. The behaviour and underlying physiology has been studied in some depth showing that the cricket auditory system solves this complex problem in a unique manner. We present an analogue very large scale integrated (aVLSI) circuit model of this process and show that results from testing the circuit agree with simulation and what is known from the behaviour and physiology of the cricket auditory system. The aVLSI circuitry is now being extended to use on a robot along with previously modelled neural circuitry to better understand the complete sensorimotor pathway.

## 1 Introduction

Understanding how insects carry out complex sensorimotor tasks can help in the design of simple sensory and robotic systems. Often insect sensors have evolved into intricate filters matched to extract highly specific data from the environment which solves a particular problem directly with little or no need for further processing [1]. Examples include head stabilisation in the fly, which uses vision amongst other senses to estimate self-rotation and thus to stabilise its head in flight, and phonotaxis in the cricket.

Because of the narrowness of the cricket body (only a few millimetres), the Interaural Time Difference (ITD) for sounds arriving at the two sides of the head is very small (10–20μs). Even with the tympanal membranes (eardrums) located, as they are, on the forelegs of the cricket, the ITD only reaches about 40μs, which is too low to detect directly from timings of neural spikes. Because the wavelength of the cricket calling song is significantly greater than the width of the cricket body the Interaural Intensity Difference (IID) is also very low. In the absence of ITD or IID information, the cricket uses phase to determine direction. This is possible because the male cricket produces an almost pure tone for its calling song.

---

[*]School of Electrical and Information Engineering,
[+]Institute of Perception, Action and Behaviour.

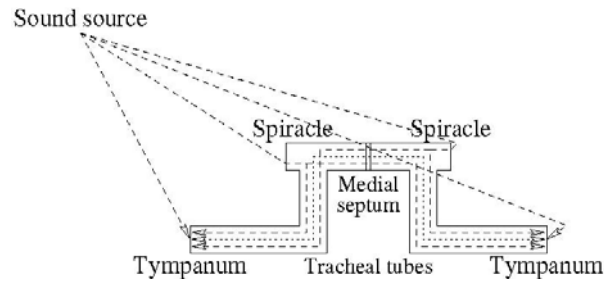

Figure 1: The cricket auditory system. Four acoustic inputs channel sounds directly or through tracheal tubes onto two tympanal membranes. Sound from contralateral inputs has to pass a (double) central membrane (the medial septum), inducing a phase delay and reduction in gain. The sound transmission from the contralateral tympanum is very weak, making each eardrum effectively a 3 input system.

The physics of the cricket auditory system is well understood [2]; the system (see Figure 1) uses a pair of sound receivers with four acoustic inputs, two on the forelegs, which are the external surfaces of the tympana, and two on the body, the prothoracic or acoustic spiracles [3]. The connecting tracheal tubes are such that interference occurs as sounds travel inside the cricket, producing a directional response at the tympana to frequencies near to that of the calling song. The amplitude of vibration of the tympana, and hence the firing rate of the auditory afferent neurons attached to them, vary as a sound source is moved around the cricket and the sounds from the different inputs move in and out of phase. The outputs of the two tympana match when the sound is straight ahead, and the inputs are bilaterally symmetric with respect to the sound source. However, when sound at the calling song frequency is off-centre the phase of signals on the closer side comes better into alignment, and the signal increases on that side, and conversely decreases on the other. It is that crossover of tympanal vibration amplitudes which allows the cricket to track a sound source (see Figure 6 for example).

A simplified version of the auditory system using only two acoustic inputs was implemented in hardware [4], and a simple 8-neuron network was all that was required to then direct a robot to carry out phonotaxis towards a species-specific calling song [5].

A simple simulator was also created to model the behaviour of the auditory system of Figure 1 at different frequencies [6]. Data from Michelsen et al. [2] (Figures 5 and 6) were digitised, and used together with average and "typical" values from the paper to choose gains and delays for the simulation. Figure 2 shows the model of the internal auditory system of the cricket from sound arriving at the acoustic inputs through to transmission down auditory receptor fibres. The simulator implements this model up to the summing of the delayed inputs, as well as modelling the external sound transmission.

Results from the simulator were used to check the directionality of the system at different frequencies, and to gain a better understanding of its response. It was impractical to check the effect of leg movements or of complex sounds in the simulator due to the necessity of simulating the sound production and transmission. An aVLSI chip was designed to implement the same model, both allowing more complex experiments, such as leg movements to be run, and experiments to be run in the real world.

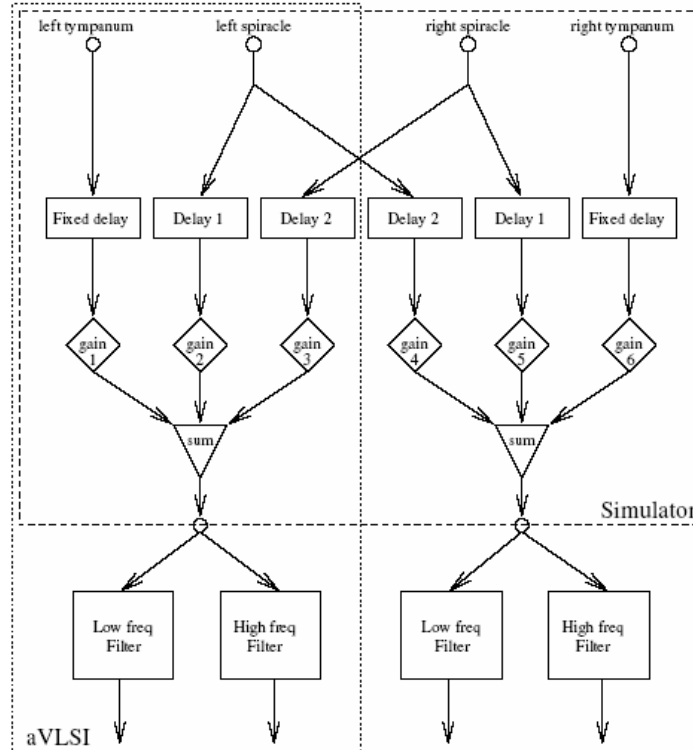

Figure 2: A model of the auditory system of the cricket, used to build the simulator and the aVLSI implementation (shown in boxes).

These experiments with the simulator and the circuits are being published in [6] and the reader is referred to those papers for more details. In the present paper we present the details of the circuits used for the aVLSI implementation.

## 2  Circuits

The chip, implementing the aVLSI box in Figure 2, comprises two all-pass delay filters, three gain circuits, a second-order narrow-band band-pass filter, a first-order wide-band band-pass filter, a first-order high-pass filter, as well as supporting circuitry (including reference voltages, currents, *etc.*). A single aVLSI chip (MOSIS tiny-chip) thus includes half the necessary circuitry to model the complete auditory system of a cricket. The complete model of the auditory system can be obtained by using two appropriately connected chips.

Only two all-pass delay filters need to be implemented instead of three as suggested by Figure 2, because it is only the relative delay between the three pathways arriving at the one summing node that counts. The delay circuits were implemented with fully-differential gm-C filters. In order to extend the frequency range of the delay, a first-order all-pass delay circuit was cascaded with a second-order all-pass delay circuit. The resulting addition of the first-order delay and the second-order delay allowed for an approximately flat delay response for a wider bandwidth as the decreased delay around the corner frequency of the first-order filter cancelled with the increased delay of the second-order filter around its resonant frequency. Figure 3 shows the first- and second-order sections of the all-pass delay circuit. Two of these

circuits were used and, based on data presented in [2], were designed with delays of 28µs and 62µs, by way of bias current manipulation. The operational trans-conductance amplifier (OTA) in figure 3 is a standard OTA which includes the common-mode feedback necessary for fully differential designs. The buffers (Figure 3) are simple, cascoded differential pairs.

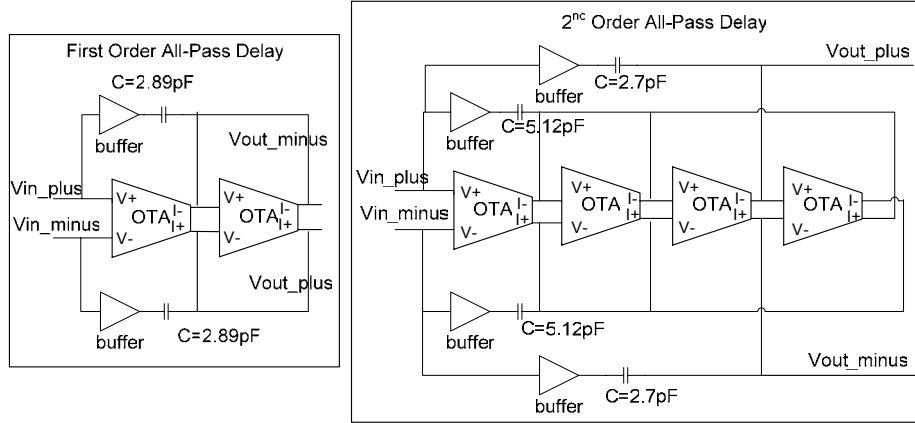

Figure 3: The first-order all-pass delay circuit (left) and the second-order all-pass delay (right).

The differential output of the delay circuits is converted into a current which is multiplied by a variable gain implemented as shown in Figure 4. The gain cell includes a differential pair with source degeneration via transistors N4 and N5. The source degeneration improves the linearity of the current. The three gain cells implemented on the aVLSI have default gains of 2, 3 and 0.91 which are set by holding the *default* input high and appropriately ratioing the bias currents through the value of *vbiasp*. To correct any on-chip mismatches and/or explore other gain configurations a current splitter cell [7] (*p-splitter*, figure 4) allows the gain to be programmed by digital means post fabrication. The current splitter takes an input current (*Ibias*, figure 4) and divides it into branches which recursively halve the current, i.e., the first branch gives ½ *Ibias*, the second branch ¼ *Ibias*, the third branch 1/8 *Ibias* and so on. These currents can be used together with digitally controlled switches as a Digital-to-Analogue converter. By holding *default* low and setting *C5:C0* appropriately, any gain – from 4 to 0.125 – can be set. To save on output pins the program bits (*C5:C0*) for each of the three gain cells are set via a single 18-bit shift register in bit-serial fashion.

Summing the output of the three gain circuits in the current domain simply involves connecting three wires together. Therefore, a natural option for the filters that follow is to use current domain filters. In our case we have chosen to implement log-domain filters using MOS transistors operating in weak inversion. Figure 5 shows the basic building blocks for the filters – the Tau Cell [8] and the multiplier cell – and block diagrams showing how these blocks were connected to create the necessary filtering blocks. The Tau Cell is a log-domain filter which has the first-order response:

$$\frac{I_{out}}{I_{in}} = \frac{1}{s\tau + 1}, \quad \text{where } \tau = \frac{nC_aV_T}{I_a}$$

and $n$ = the slope factor, $V_T$ = thermal voltage, $C_a$ = capacitance, and $I_a$ = bias current. In figure 5, the input currents to the Tau Cell, $I_{mult}$ and $A*I_a$, are only used

when building a second-order filter. The multiplier cell is simply a translinear loop where: $I_{out1} * I_{mult} = I_{out2} * AI_a$ or $I_{mult} = AI_a I_{out2}/I_{out1}$. The configurations of the Tau Cell to get particular responses are covered in [8] along with the corresponding equations. The high frequency filter of Figure 2 is implemented by the high-pass filter in Figure 5 with a corner frequency of 17kHz. The low frequency filter, however, is divided into two parts since the biological filter's response (see for example Figure 3A in [9]) separates well into a narrow second-order band-pass filter with a 10kHz resonant frequency and a wide band-pass filter made from a first-order high-pass filter with a 3kHz corner frequency followed by a first-order low-pass filter with a 12kHz corner frequency. These filters are then added together to reproduce the biological filter. The filters' responses can be adjusted post fabrication via their bias currents. This allows for compensation due to processing and matching errors.

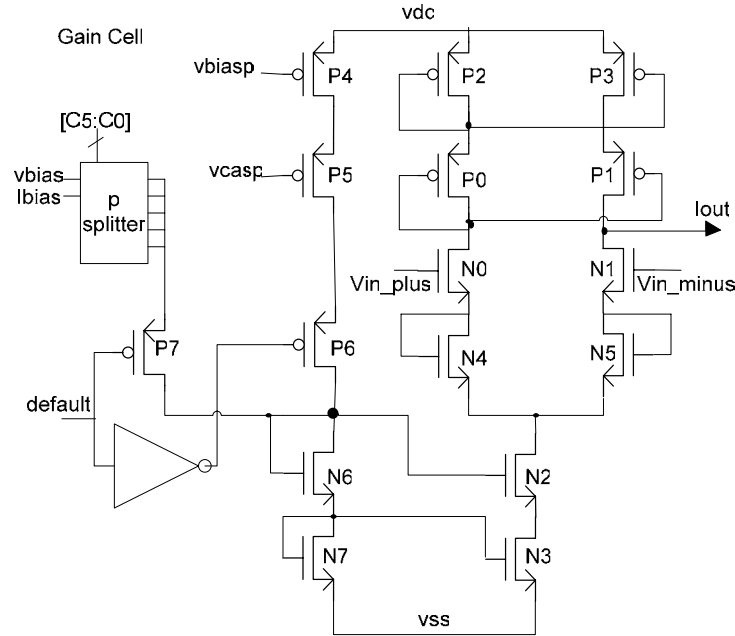

Figure 4: The Gain Cell above is used to convert the differential voltage input from the delay cells into a single-ended current output. The gain of each cell is controllable via a programmable current cell (p_splitter).

An on-chip bias generator [7] was used to create all the necessary current biases on the chip. All the main blocks (delays, gain cells and filters), however, can have their on-chip bias currents overridden through external pins on the chip.

The chip was fabricated using the MOSIS AMI 1.6μm technology and designed using the Cadence Custom IC Design Tools (5.0.33).

## 3   Methods

The chip was tested using sound generated on a computer and played through a soundcard to the chip. Responses from the chip were recorded by an oscilloscope, and uploaded back to the computer on completion. Given that the output from the

chip and the gain circuits is a current, an external current-sense circuit built with discrete components was used to enable the output to be probed by the oscilloscope.

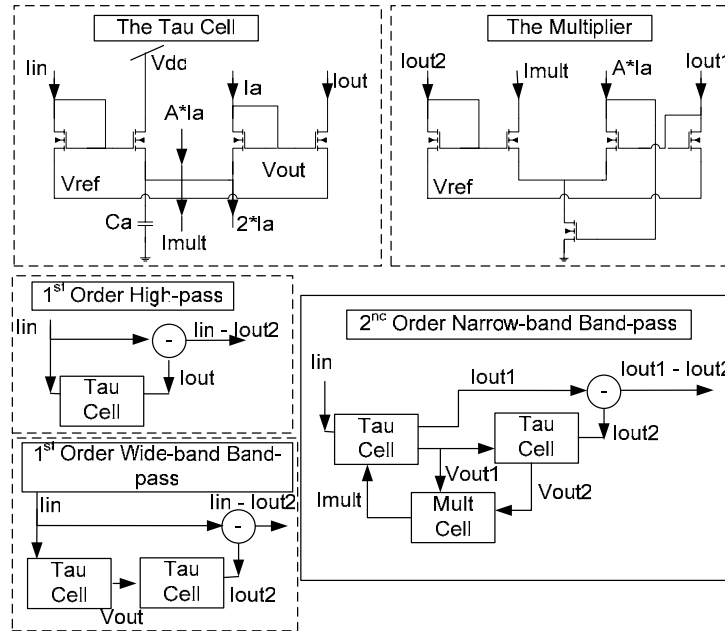

Figure 5: The circuit diagrams for the log-domain filter building blocks – The Tau Cell and The Multiplier – along with the block diagrams for the three filters used in the aVLSI model.

Initial experiments were performed to tune the delays and gains. After that, recordings were taken of the directional frequency responses. Sounds were generated by computer for each chip input to simulate moving the forelegs by delaying the sound by the appropriate amount of time; this was a much simpler solution than using microphones and moving them using motors.

## 4   Results

The aVLSI chip was tested to measure its gains and delays, which were successfully tuned to the appropriate values. The chip was then compared with the simulation to check that it was faithfully modelling the system. A result of this test at 4kHz (approximately the cricket calling-song frequency) is shown in Figure 6. Apart from a drop in amplitude of the signal, the response of the circuit was very similar to that of the simulator. The differences were expected because the aVLSI circuit has to deal with real-world noise, whereas the simulated version has perfect signals. Examples of the gain versus frequency response of the two log-domain band-pass filters are shown in Figure 7. Note that the narrow-band filter peaks at 6kHz, which is significantly above the mating song frequency of the cricket which is around 4.5kHz. This is not a mistake, but is observed in real crickets as well. As stated in the introduction, a range of further testing results with both the circuit and the simulator are being published in [6].

# 5 Discussion

The aVLSI auditory sensor in this research models the hearing of the field cricket *Gryllus bimaculatus*. It is a more faithful model of the cricket auditory system than was previously built in [4], reproducing all the acoustic inputs, as well as the responses to frequencies of both the co specific calling song and bat echolocation chirps. It also generates outputs corresponding to the two sets of behaviourally relevant auditory receptor fibres. Results showed that it matched the biological data well, though there were some inconsistencies due to an error in the specification that will be addressed in a future iteration of the design. A more complete implementation across all frequencies was impractical because of complexity and size issues as well as serving no clear behavioural purpose.

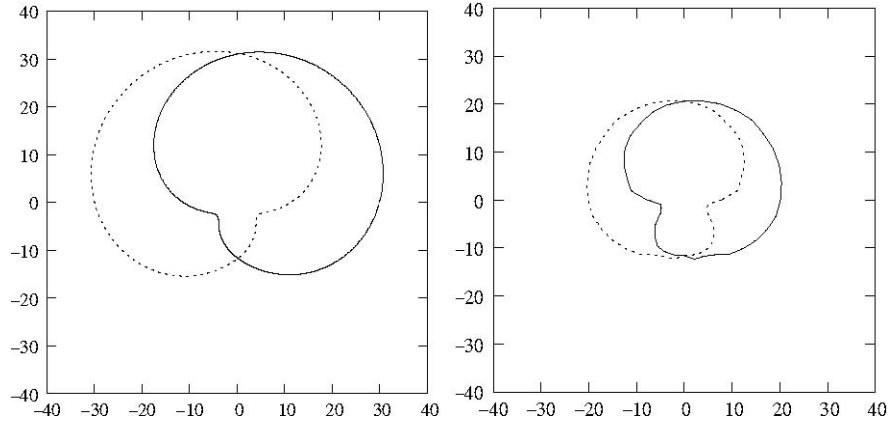

Figure 6: Vibration amplitude of the left (dotted) and right (solid) virtual tympana measured in decibels in response to a 4kHz tone in simulation (left) and on the aVLSI chip (right). The plot shows the amplitude of the tympanal responses as the sound source is rotated around the cricket.

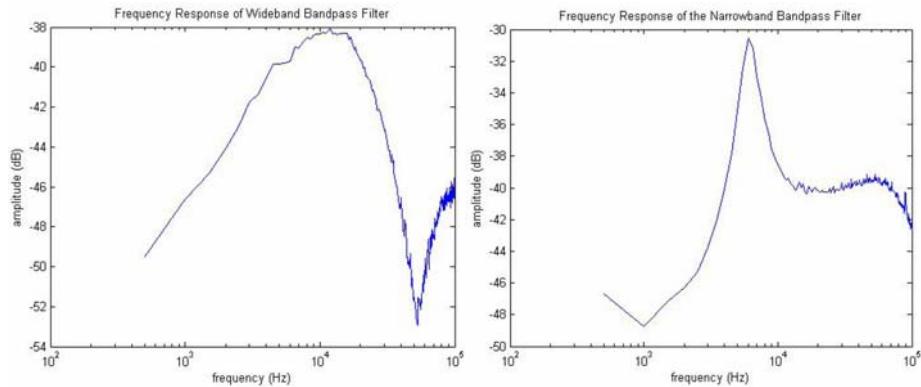

Figure 7: Frequency-Gain curves for the narrow-band and wide-band band-pass filters.

The long-term aim of this work is to better understand simple sensorimotor control loops in crickets and other insects. The next step is to mount this circuitry on a robot to carry out behavioural experiments, which we will compare with existing and new behavioural data (such as that in [10]). This will allow us to refine our models of the neural circuitry involved. Modelling the sensory afferent neurons in hardware is necessary in order to reduce processor load on our robot, so the next revision will

include these either onboard, or on a companion chip as we have done before [11]. We will also move both sides of the auditory system onto a single chip to conserve space on the robot.

It is our belief and experience that, as a result of this intelligent pre-processing carried out at the sensor level, the neural circuits necessary to accurately model the behaviour will remain simple.

## Acknowledgments

The authors thank the Institute of Neuromorphic Engineering and the UK Biotechnology and Biological Sciences Research Council for funding the research in this paper.

## References

[1] R. Wehner. Matched filters – neural models of the external world. J Comp Physiol A, 161: 511–531, 1987.

[2] A. Michelsen, A. V. Popov, and B. Lewis.Physics of directional hearing in the cricket Gryllus bimaculatus. Journal of Comparative Physiology A, 175:153–164, 1994.

[3] A. Michelsen. The tuned cricket. News Physiol. Sci., 13:32–38, 1998.

[4] H. H. Lund, B. Webb, and J. Hallam. A robot attracted to the cricket species Gryllus bimaculatus. In P. Husbands and I. Harvey, editors, Proceedings of 4th European Conference on Artificial Life, pages 246–255. MIT Press/Bradford Books, MA., 1997.

[5] R Reeve and B. Webb. New neural circuits for robot phonotaxis. Phil. Trans. R. Soc. Lond. A, 361:2245–2266, August 2003.

[6] R. Reeve, A. van Schaik, C. Jin, T. Hamilton, B. Torben-Nielsen and B. Webb Directional hearing in a silicon cricket. *Biosystems*, (in revision), 2005b

[7] T. Delbrück and A. van Schaik, Bias Current Generators with Wide Dynamic Range, Analog Integrated Circuits and Signal Processing 42(2), 2005

[8] A. van Schaik and C. Jin, The Tau Cell: A New Method for the Implementation of Arbitrary Differential Equations, IEEE International Symposium on Circuits and Systems (ISCAS) 2003

[9] Kazuo Imaizumi and Gerald S. Pollack. Neural coding of sound frequency by cricket auditory receptors. The Journal of Neuroscience, 19(4):1508–1516, 1999.

[10]Berthold Hedwig and James F.A. Poulet. Complex auditory behaviour emerges from simple reactive steering. Nature, 430:781–785, 2004.

[11]R. Reeve, B. Webb, A. Horchler, G. Indiveri, and R. Quinn. New technologies for testing a model of cricket phonotaxis on an outdoor robot platform. *Robotics and Autonomous Systems*, 51(1):41-54, 2005.
